# D2R2: Diffusion-based Representation with Random Distance Matching for Tabular Few-shot Learning

**Ruoxue Liu** [*]
HKUST
rliuaj@connect.ust.hk

**Linjiajie Fang** [*]
HKUST
lfangad@connect.ust.hk

**Wenjia Wang** [†]
HKUST (GZ) and HKUST
wenjiawang@ust.hk

**Bing-Yi Jing** [†]
SUSTech
jingby@sustech.edu.cn

## Abstract

Tabular data is widely utilized in a wide range of real-world applications. The challenge of few-shot learning with tabular data stands as a crucial problem in both industry and academia, due to the high cost or even impossibility of annotating additional samples. However, the inherent heterogeneity of tabular features, combined with the scarcity of labeled data, presents a significant challenge in tabular few-shot classification. In this paper, we propose a novel approach named Diffusion-based Representation with Random Distance matching (D2R2) for tabular few-shot learning. D2R2 leverages the powerful expression ability of diffusion models to extract essential semantic knowledge crucial for denoising process. This semantic knowledge proves beneficial in few-shot downstream tasks. During the training process of our designed diffusion model, we introduce a random distance matching to preserve distance information in the embeddings, thereby improving effectiveness for classification. During the classification stage, we introduce an instance-wise iterative prototype scheme to improve performance by accommodating the multimodality of embeddings and increasing clustering robustness. Our experiments reveal the significant efficacy of D2R2 across various tabular few-shot learning benchmarks, demonstrating its state-of-the-art performance in this field.

## 1 Introduction

Learning with a large set of data with only a few labeled samples is an essential requirement for industry and academia, primarily due to the high cost of annotating samples. However, accurately classifying new data with a scantily labeled training set and an unlabeled training set poses a formidable challenge, as the effectiveness of statistical and modern deep learning systems in supervised learning heavily relies on the large size of the labeled set [36]. This underscores the need for research in few-shot learning in situations where labeled data is scarce. For example, in one-shot learning, each class of the training set contains only one labeled sample, and the goal is to classify new test samples. While few-shot learning has garnered considerable attention in computer vision (CV) [9, 21] and natural language processing (NLP) [30, 31], it remains relatively under-explored in the field of tabular data. Nevertheless, few-shot learning holds great importance in the context of tabular data, as limited labeled tabular data is inherently common in many real-world applications, such as fraud detection [7], disease diagnosis [40], and social science [27]. However, modeling with limited labeled tabular data presents three significant challenges.

---

[*]Equal contribution. † Corresponding authors.

The first challenge is due to the scarcity of labeled training samples, which leads to a deficient understanding of the test class. In this case, semantic knowledge, encompassing general and low-frequency information not tied to any particular task, can be highly beneficial in few-shot learning [39, 52, 56]. This is because representing a class with limited labeled samples inherently involves ambiguity, necessitating the use of semantic knowledge acquired from unlabeled ones to refine the class definition and address this ambiguity. Although for CV [21] and NLP [31], semantic knowledge can be effectively derived based on the learned spatial structure patterns between pixels or tokens, it is more challenging to extract semantic knowledge for tabular data, since they typically lack local relationships between columns.

The second challenge arises from the diverse characteristics of tabular features. Tabular data comprises numerical and categorical features. Numerical features contain ordinal values but have multiple modes, while categorical features present distinct and incomparable values. Many existing statistical and deep neural network models struggle to effectively handle these mixed data types. This challenge underscores the importance of simultaneously modeling continuous and categorical features.

The third challenge pertains to the utilization of prototype classification with tabular data. Numerous approaches suggest employing average embeddings of the labeled support set for classification [49, 11, 28]. However, the limited labeled support samples lead to large variance and imprecision of the prototypes, highlighting the need for more information from unlabeled samples. Additionally, embeddings from the same class may exhibit multimodal behavior (see Figure 3 for examples), which means that using the average embeddings of one class could result in inaccurate classification results.

In addressing the aforementioned challenges of few-shot learning, significant efforts have been made over the past decade. One approach employs a meta-learning scheme. A common method is to generate a pseudo-label for each unlabeled data and then train the model using pseudo-label [25, 43, 28]. Other approaches utilize self-generated tasks, such as CACTUs [19], UMTRA [22] and Meta-GMVAE [24]. However, despite the effectiveness of prior works on image datasets, we find that applying each method to the tabular domain is highly non-trivial. These methods often assume uniformity in feature types and rely on strong spatial and sequential relationships among features, which do not hold for tabular data. For instance, augmentation techniques are readily applicable to images, where spatial relationships between pixels can be leveraged, as shown by UMTRA [22]. However, such techniques are difficult to apply to tabular data, which lacks these structural patterns [4]. STUNT [28] is designed for tabular data using a proxy-based approach. However, this methods may struggle to acquire useful semantic knowledge due to the disparities between the constructed tasks and the target task, leading to diminished performance for testing. Moreover, it applies the same methods to both numerical and categorical features, neglecting the unique information in different types of data. This oversight hampers the model's effectiveness. Recent studies have showed that self-supervised learning can leverage unlabeled datasets to acquire semantic knowledge and transferable representations of images [16] and languages [14]. However, these approaches heavily rely on augmentation schemes. For tabular data, the heterogeneous feature types pose challenges for augmentation, and only a few methods have been proposed for tabular data. State-of-the-art methods along this line include VIME [55], SCARF [4], SubTab [47].

In this work, we propose a novel framework named Diffusion-based Representation with Random Distance Matching (D2R2) for tabular few-shot learning. D2R2 extracts representations by training a designed conditional diffusion model and aligning the distances from various random projection spaces. Subsequently, we introduce the instance-wise iterative prototype to conduct few-shot classification, which addresses the multimodal behavior of embeddings and enables the creation of a robust classifier. Finally, we design an unsupervised validation scheme to address the absence of the labeled validation set for hyperparameter selection. Different from meta-learning and self-supervised learning methods, D2R2 does not relies on self-generated tasks or augmentation schemes, but instead uses a general training process. The D2R2 framework offers the following advantages:

Considering the first challenge mentioned earlier, we avoid to rely on proxy methods or augmentation methods, which are limited by the lack of local relationships in tabular data. Instead, we create an information bottleneck for extracting semantic knowledge, named D2R2, which leverages the strong expressiveness of the diffusion model and distance information from pairwise comparison. Firstly, we modify the diffusion model to serve as a representation extractor. The powerful expressiveness of diffusion model enables the extraction of semantic knowledge crucial to denoising. And the information bottleneck induced by conditional information, obtains low-frequency information over

details. This approach outperforms alternative representation learning and meta-learning methods, as demonstrated by our experiments. Moreover, the representations exhibit robustness due to the introduction of designed diffusion noise schedule and subsequent denoising process, which reinforces their stability against tabular column perturbations.

Nevertheless, relying solely on diffusion-based representations may not provide sufficient distance information to build a subsequent prototype classifier, which classifies test samples based on their distance between prototypes. Hence, we introduce a random distance matching (RDM) loss during the training of the diffusion model to obtain the distance information. One intuition of applying RDM is that if the class structure can be characterized by certain distance information, the embeddings should preserve such distance structure. Another intuition is that if two samples belong to the same class in the ground truth, their embeddings should appear similar from any perspective. Random distance matching achieves these ideas by projecting data into diverse random spaces, which encompass class structure information. Consequently, if two data points are close in multiple random spaces, they are highly likely to belong to the same class. Besides, two distinct Random Distance matching are designed for continuous and discrete features based on their characteristics, respectively. In this way, we can simultaneously model numerical and categorical features within one model, addressing the second challenge mentioned earlier. Overall, our newly designed D2R2 training process not only extracts high-quality semantic knowledge for few-shot learning but also reveals distance information for prototype classification, while remaining adaptive to mixture types of tabular data.

Moreover, during the classification phase, we predict the class of a test sample to be the same as the class of the nearest prototype. Given that the embedding of a single class may demonstrate multimodality, we introduce a novel instance-wise iterative prototype. This approach tackles the issue of multimodal behavior by creating several prototypes within one class and improving the robustness of prototypes by iterative refinement, resolving the third challenge previously mentioned.

Lastly, to address the absence of a labeled validation set in unsupervised learning for hyperparameter selection, we devise a validation scheme by generating pseudo-labels for the unlabeled dataset using soft k-means clustering of raw features.

Our contribution is summarized as follows:

- To the best of our knowledge, we are the first to propose a specifically designed diffusion method to learn semantic knowledge for tabular data.

- We propose an innovative framework, D2R2, to extract representations in tabular few-shot learning, which is built upon the designed diffusion process and random distance matching. D2R2 not only captures high-quality semantic knowledge, but also incorporates distance information for the subsequent prototype classifier. Moreover, it adapts well to mixture types of tabular data.

- To further improve few-shot classification performance, we introduce a novel classifier with instance-wise iteration prototypes. This classifier is able to construct highly accurate and stable prototypes, while also revealing the multimodal behavior of a single class.

- We conduct extensive experiments to evaluate our framework for tabular few-shot learning, comparing our method with 15 state-of-the-art literature baselines across nine datasets. Among various tabular datasets, D2R2 outperforms other baselines by a significant margin.

## 2   Related work

**Supervised and Semi-supervised learning**. Some supervised classifiers have strong ability to learn from limited samples. Methods along this line include CatBoost [35], TabPFN [18], a transformer-based network designed to make predictions on a small tabular dataset; k-nearest neighbor classifier (kNN) [32], the nearest neighbor prototype classifier. However, they are inadequate in few-shot learning because they are not intended to learn the semantic knowledge. Semi-supervised learning frameworks are designed to improve model generalization by creating a strong connection between limited labeled samples and unlabeled samples, including Mean Teacher (MT) [46], Interpolation Consistency Training (ICT) [48], and Meta Pseudo Labels (MPL) [33]. In semi-supervised learning, a model trains on labeled data and then predicts labels for unlabeled data. However, this method requires more labeled data than few-shot learning contexts.

**Few-shot meta-learning**. Few-shot learning aims to train models to adapt to downstream tasks with minimal labeled examples. This can be accomplished by employing meta-learning techniques on related tasks, enabling the acquisition of prior knowledge that can be utilized to solve new tasks [50]. Existing deep few-shot methods can be divided into three main categories. First, optimization-based meta-learning methods [5, 15, 26, 29] devise a proficient optimization strategy that adapts to downstream tasks effectively. Second, metric-based methods [13, 30, 34, 51] concentrate on latent space to derive meaningful feature embeddings, and then make predictions based on the similarity between support and query embeddings. Third, data generative strategies emphasize creating more varied samples to train a more precise classifier. Most research on few-shot meta-learning focuses on NLP and CV tasks, such as CACTUs [19] and UMTRA [22], while a small subset tackles tabular few-shot learning. STUNT [28] meta-learns generalizable knowledge from self-generated tasks on an unlabeled tabular training set. However, these methods may generate ineffective semantic knowledge because of the gap between the self-generated tasks and the testing task.

**Self-supervised learning**. Our research addresses scenarios where an unlabeled training set and a small labeled support set ($K$-shot) are used to predict class labels for a test query set, as detailed in Section 3 of our paper. Self-supervised learning is particularly effective at developing robust representations from unlabeled data [8, 37]. These approaches focus on pre-training the representation by utilizing domain-specific inductive biases, like the spatial relationships in images. Notably, prior research has demonstrated the effectiveness of self-supervised learning in few-shot scenarios relative to meta-learning techniques [12]. Chen et al. [10] added an autoencoder in the diffusion process and Yang et al.[53] introduces a latent Denoising Autoencoder architecture where the learned representations are used for Denoising Autoencoder, but the Autoencoder reconstruction process is less suitable for tabular data as discussed in STUNT [28]. Other works on self-supervised learning schemes rely on augmentation schemes. It is unclear how to extend such methods to the tabular domain due to the heterogeneous characteristics of tabular datasets. Moreover, for tabular data, multimodal and categorical features make augmentation difficult, and few effective methods have been proposed in tabular data setting. Recent state-of-the-art tabular learning augmentation techniques include masking cell: VIME [55], constructing subsets: SubTab [47], and contrastive learning: SCARF [4]. However, they fail to deliver substantial performance enhancements for few-shot tabular learning in our experiments. Rather, we train the unlabeled dataset through an unsupervised diffusion framework that does not rely on the effect of augmentation.

# 3 Problem definition

In this paper, we explore few-shot learning for tabular classification.Tabular data refers to the dataset organized in tables, which is a structured format that presents information in rows and columns. Such data can be represented as $D = \{\boldsymbol{x}_i\}_{i=1}^n \subset \mathcal{R}^d$ consisting of $n$ instances and $d$ dimensional features. Each data instance $\boldsymbol{x}_i = (x_i^1, x_i^2, ..., x_i^d)$ may or may not hold strong relationship among features. The features in tabular data typically vary, comprising both numerical and categorical labels.

Our study on tabular few-shot learning focus on a very typical scenario in this field, adhering to the definition outlined in STUNT [28]. We have an unlabeled training set $\mathcal{D}_u = \{\mathbf{x}_i^u\}_{i=1}^{N_u}$ and a limited labeled support set $\mathcal{S} = \{(\mathbf{x}_i^s, y_i^s)\}_{i=1}^{N_S}$, $\mathbf{x}_i^s \in \mathbb{R}^d$ and $y_i^s \in \{1, 2, ..., C\}$ represent inputs and class labels, respectively. We assume that $N_u \gg N_S$. Our goal is to predict the class labels of testing query set $\mathcal{Q} = \{\mathbf{x}_i^q\}_{i=1}^{N_Q}$. In the $N$-way $K$-shot setting, the classification is conducted with $N$ targeted classes and each class in the support set has $K$ labeled samples. Such scenarios are common in critical applications like credit risk assessment [7] and diagnosing patients with rare diseases [40].

# 4 Methodology

In this section, we introduce the overall design of the framework Diffusion-based Representation learning with Random Distance matching (D2R2) in detail. In order to leverage the semantic knowledge extraction capabilities of diffusion models, D2R2 employs a conditional diffusion model on the unlabeled dataset to learn an embedding space (Section 4.1). In order to enhance the clustering ability of the embedding space, facilitating subsequent classification, we modify the training process of the diffusion model, in order to ensure that the distances between instances align across the embedding space and various projected spaces. (Section 4.2). Figure 1 illustrates aforementioned

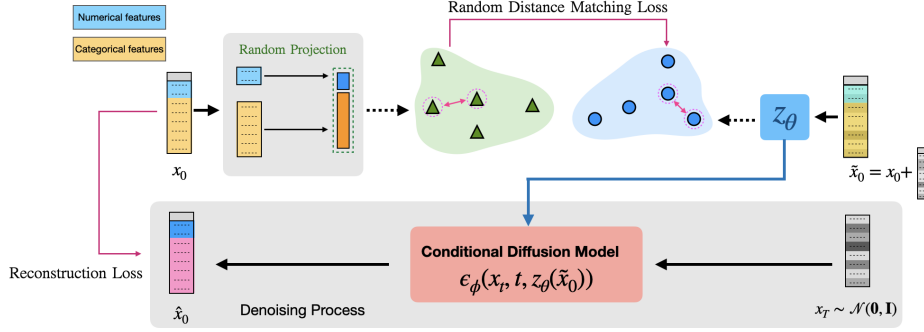

Figure 1: The diagram depicts the training process of the embedding space. Specifically, after a noise perturbation, each instance undergoes mapping via an embedding function $z_\theta$. The embedding is then incorporated into a conditional diffusion model for noise prediction. The parameters of the noise model $\epsilon_\phi$ and the embedding function $z_\theta$ are concurrently optimized using the reconstruction loss. Additionally, the instance is subjected to mapping through a random linear projection into an alternate metric space. Two distinct Random projections are generated for numerical and categorical features, respectively. We align the distances in the embedding space and the random projection space using the RDM loss to ensure that the embedding function effectively preserves distance information, which is beneficial for downstream classification tasks.

training process. In the classification stage, we construct a novel instance-wise iterative classifier to predict the testing samples on the learned embedding space (Section 4.3). Finally, due to the absence of a validation set for above unsupervised embedding space learning, we propose to use pseudo-label validation on the unlabeled dataset to select hyperparameters (Section 4.4). A summary of the training algorithm of D2R2 is presented in Appendix A.

## 4.1 Diffusion-based representation Learning

Diffusion-based generative models [42, 17, 44, 45] are latent variable models that use a Markovian noising and parameterized denoising process to model the data distribution thus generate realistic samples. Details of diffusion models are provided in the appendix B. The diffusion process for a given sample $\mathbf{x}_0$ is defined as $\mathbf{x}_t = \sqrt{\bar{\alpha}_t}\mathbf{x}_0 + \sqrt{1 - \bar{\alpha}_t}\epsilon$, where $\epsilon \sim \mathcal{N}(0, \mathbf{I})$, $\bar{\alpha}_t = \prod_{t'=1}^{t}(1 - \beta'_t)$, $\beta_t$ is a pre-defined variance schedule, and the timestep $t$ is known. The true noise is $\epsilon(\mathbf{x}_t, t, \mathbf{x}_0) = 1/\sqrt{1 - \bar{\alpha}_t}(\mathbf{x}_t - \sqrt{\bar{\alpha}_t}\mathbf{x}_0)$. Following DDPMs [17], the denoising model can be implemented by neural networks with learnable parameters $\phi$ by directly predicting the noise $\epsilon$:

$$\mathcal{L}_{\text{DDPM}}(\phi) = \mathbf{E}_{t,\mathbf{x}_0}||\epsilon_\phi(\mathbf{x}_t, t) - \epsilon||_2^2, \tag{1}$$

Diffusion models can be extended to conditional generative models $\epsilon_\phi(\mathbf{x}_t, t, c)$ by inserting the conditional information $c$ to generate specific class samples [45, 1].

We now present the diffusion-based representation learning part of D2R2. We employ the diffusion models in a distinct way different from generation. We modify the diffusion model and repurposing it to act as representation extractor. Specifically, we represent the trainable embedding function as $z_\theta : \mathbb{R}^d \to \mathbb{R}^p$, where $z_\theta(\mathbf{x}_0)$ maps the noiseless input $\mathbf{x}_0$ to the latent space of dimension $p$. Formally, the reconstruction loss for the diffusion-based representation learning is

$$\mathcal{L}_{\text{recon}}(\phi, \theta) = \mathbf{E}_{t,\mathbf{x}_0,\mathbf{x}_t}||\epsilon_\phi(\mathbf{x}_t, t, z_\theta(\tilde{\mathbf{x}}_0)) - \epsilon||_2^2, \tag{2}$$

where $\epsilon_\phi$ is a trainable noise model, and $\tilde{\mathbf{x}}_0$ is an augmentation version of the noiseless data input $\mathbf{x}_0$. In our method, we use Gaussian noise perturbation $\tilde{\mathbf{x}}_0 = \mathbf{x}_0 + \sigma \mathcal{N}(0, 1)$. For numerical features we set $\sigma_i$ to be the standard deviation of numerical features in $\mathcal{D}_u$, and for categorical (one-hot) features, we use a fixed $\sigma_i$ to smooth the one-hot indicator values.

Reasons that the designed diffusion model can extract semantic knowledge come from twofold. Firstly, the diffusion model with powerful expressiveness encodes the information needed for denoising. Specifically, in conditional diffusion models, the noise reconstruction loss $\mathbf{E}_{t,\mathbf{x}_0,\mathbf{x}_t}||\epsilon_\phi(\mathbf{x}_t, t, c) - \epsilon||_2^2$ trains the noise prediction function $\epsilon_\phi(\mathbf{x}_t, t, c)$ to predict the true noise $\epsilon(\mathbf{x}_t, t, \mathbf{x}_0)$ given the noisy

sample $\mathbf{x}_t$, the knowing $t$ and the condition information $c$. If $c = \mathbf{x}_0$, we could expect $\epsilon_\phi$ can almost perfectly recover $\epsilon$. By replacing the conditional information $c$ by a function $z_\theta(\mathbf{x}_0)$ that maps to an embedding space with lower dimension than $\mathbf{x}_0$, we introduce an information bottleneck to the noise reconstruction process. This forces $z_\theta$ to extract effective information for denoising from $\mathbf{x}_0$, leading to representation learning through the noise reconstruction loss (2) .

From another perspective, it is pointed out the noise reconstruction loss (2) can be expressed equivalently as the weighted noisy score matching loss $\mathbf{E}_{t,\mathbf{x}_0,\mathbf{x}_t}[\lambda(t)||s_\phi - \nabla_{\mathbf{x}_t} \log p_{\sigma(t)}(\mathbf{x}_t|\mathbf{x}_0)||_2^2]$ [1], where the weights $\lambda(t) = \sigma^2(t)$ are determined by the noise scale $\sigma(1) < \sigma(2) < ... < \sigma(T)$. The choice of noise scale $\sigma(t)$ controls the granularity of the embedding function. We focus on larger timesteps thus extract the low-frequency semantic information rather than details.

## 4.2 Random distance matching

The acquired latent representation $z_\theta$ from equation (2) is utilized for subsequent classification tasks, which heavily rely on distance information between the clusters within the embedding space. Although $z_\theta(\mathbf{x}_0)$ learned through conditional diffusion models already exhibits clustering properties [1] to a certain extent, the learned embeddings do not prioritize capturing distance information between class structures, potentially hindering the effectiveness for downstream classification tasks. Taking inspiration that representations can be learned by training neural networks to predict distances in a randomly projected space [49], we propose the random distance matching (RDM) loss to align pairwise distances between the embedding space and a randomly projected space $\mathbb{R}^r$. Here we suppose that if two samples belong to the same class in the ground truth, their embeddings should be close to each other, which is reflected by the randomly projected features from any perspectives.

Specifically, we consider the random linear projections $W \in \mathbb{R}^{r \times d}$ with each element $w_{i,j}$ sampled $i.i.d.$ from a fixed distribution. The RDM loss is as follows:

$$\mathcal{L}_{\mathrm{rdm}}(\theta) = \mathbf{E}_{\mathbf{x}_0,\mathbf{x}_0',W}||d(z_\theta(\mathbf{x}_0), z_\theta(\mathbf{x}_0')) - d(W\mathbf{x}_0, W\mathbf{x}_0')||^2, \tag{3}$$

where $\mathbf{x}_0, \mathbf{x}_0'$ are any two unlabeled data points sampled from $\mathcal{D}_u$ and $d$ is a metric. Since the dimensions and the scales of the two matching spaces might not be the same, we choose $d$ to be the cosine distance.

Furthermore, in order to handle hybrid tabular data types, which involve both numerical and categorical features, we sample random projections from various distributions according to the specific feature types. We consider samples with $d_{\mathrm{num}}$ numerical features and $d_{\mathrm{cat}}$ categorical features. For numerical features, we sample a projection $W_{\mathrm{num}} \in \mathbb{R}^{r_1 \times d_{\mathrm{num}}}$ from symmetric uniform distribution: $w_{i,j} \sim_{i.i.d.} \mathrm{Unif}(-A, A)$; for categorical features, we sample a projection $W_{\mathrm{cat}} \in \mathbb{R}^{r_2 \times d_{cat}}$ from Bernoulli distribution: $w_{i,j} \sim_{i.i.d.} \mathrm{Bernoulli}(p)$. Considering both types of features, our random linear projection is defined as:

$$W\mathbf{x}_0 := \mathrm{concat}(W_{\mathrm{num}}[\mathbf{x}_0]_{\mathrm{num}}, W_{\mathrm{cat}}[\mathbf{x}_0]_{\mathrm{cat}}), \tag{4}$$

where $[\mathbf{x}_0]_{\mathrm{num}}, [\mathbf{x}_0]_{\mathrm{cat}}$ are the numerical and categorical parts of $\mathbf{x}_0$, respectively.

Overall, we define a novel diffusion-based representation learning loss as follows:

$$\mathcal{L}_{\mathrm{D2R2}}(\phi, \theta) = \mathcal{L}_{\mathrm{recon}}(\phi, \theta) + \alpha \cdot \mathcal{L}_{\mathrm{rdm}}(\theta), \tag{5}$$

where $\alpha$ is a hyperparameter to balance the noise prediction loss and RDM loss. We train $\phi$ and $\theta$ to minimize $\mathcal{L}_{\mathrm{D2R2}}$ over the unlabeled dataset $\mathcal{D}_u$. The trained embedding function $z_\theta(\mathbf{x}_0)$ is used as the representation function for the downstream classification tasks. Loss function $\mathcal{L}_{\mathrm{D2R2}}$ considers learning of embeddings from two different angles. Diffusion models possess strong generative capabilities, compelling the reconstruction loss $\mathcal{L}_{\mathrm{recon}}(\phi, \theta)$ to ensure that the learned embedding $z_\theta$ captures high-quality semantic information, which significantly influences the data distribution. The random distance loss $\mathcal{L}_{\mathrm{rdm}}(\theta)$ adjusts the embedding to accommodate the distances between data points in a random projection space, thereby improving the clustering capability of the embedding space, rendering it suitable for subsequent classification tasks.

## 4.3 Instance-wise iterative prototype

Given the trained embedding function $z_\theta(\mathbf{x})$ and the labeled support set $\mathcal{S} = \{(\mathbf{x}_i^s, y_i^s)\}_{i=1}^{N_S}$, we define the instance-wise prototype of support samples in the embedding space as $c_i = z_\theta(\mathbf{x}_i^s)$, where $i = 1, 2, .., N_s$. We can predict the label of a query sample to be the same as the nearest prototype.

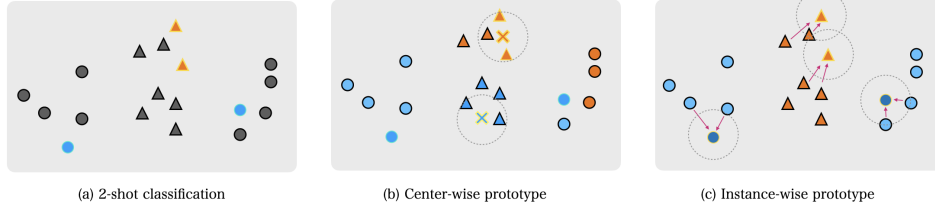

| (a) 2-shot classification | (b) Center-wise prototype | (c) Instance-wise prototype |

Figure 2: An illustration of the rationale behind instance-wise prototype. (a): In 2-shot scenarios, shapes represent the ground-truth classes in the embedding space, while the gray-colored objects await classification. (b): If the embedding of the circle class is not unimodal, averaging the prototypes leads to erroneous center suggestions and fails to classify the embeddings. (c): Considering instance-wise prototypes, each prototype contributes to the classification of nearby embeddings without generating erroneous centers.

The rationale for employing instance-wise prototypes instead of center-specific prototypes [41, 11], which averages the $K$-shot embeddings inside the $j$-th class as $c_j = \frac{1}{K} \sum_{y_i^s = j} z_\theta(\mathbf{x}_i^s)$ , is that embeddings from the same class may exhibit multimodality, while center-specific prototypes can only represent a uni-modal pattern, which might result in incorrect classification results (see Figure 2 for an illustration). On the other hand, in the few-shot learning scenarios, the scarcity of support samples brings large variance of prototypes, leading to unstable and imprecise classification outcomes. Drawing inspiration from the soft k-means algorithm [38], we leverage the weighted average of the query embeddings and support embeddings to create robust prototypes iteratively, referred to as the instance-wise iterative prototypes.

At the beginning, we initialize each prototype as the embedding of each support instance as $c_i^{(0)} = c_i = z_\theta(\mathbf{x}_i^s)$, where $i = 1, 2, .., N_s$. At the $l$-th iteration step, the probability that the query $x \in \mathcal{Q}$ belonging to the $i$-th prototype is:

$$p_\theta^{(l)}(i|x; \mathcal{S}) = \frac{\exp(-||z_\theta(\mathbf{x}) - c_i^{(l)}||_2^2 / \tau)}{\sum_{\mathbf{x}_{i'} \in \mathcal{S}} \exp(-||z_\theta(\mathbf{x}) - c_{i'}^{(l)})||_2^2 / \tau)}, \quad (6)$$

Then we update the prototypes at the $(l+1)$-th iteration based on the weighted average of query embeddings:

$$c_i^{(l+1)} = \frac{c_i^{(l)} + \sum_{x \in \mathcal{Q}} p_\theta^{(l)}(i|\mathbf{x}; \mathcal{S}) \cdot z_\theta(\mathbf{x})}{1 + \sum_{\mathbf{x} \in \mathcal{Q}} p_\theta^{(l)}(i|\mathbf{x}; \mathcal{S})}. \quad (7)$$

After the $L$-th iteration, we predict the class of a query sample $\mathbf{x}$ to be the same class of the nearest prototype:

$$\hat{y} = \arg\max_{y_i^s} p_\theta^{(L)}(i|\mathbf{x}; \mathcal{S}). \quad (8)$$

## 4.4 Pseudo-label validation

One challenge in the proposed unsupervised learning lies in the lack of a labeld validation set for hyper-parameter tuning. For example, in 1-shot classification, the only one labeled sample is used for training, with no additional labeled samples available for validation. We tackle this problem by generating pseudo labels for the unlabeled dataset using soft k-means of raw features.

Formally, we sample a validation set $\mathcal{D}_{\text{val}} \subset \mathcal{D}_u$. During each validation process, we randomly sample $K'$ points $\{\mathbf{x}_k\}_{k=1}^{K'} \subset \mathcal{D}_{\text{val}}$, which are regarded as pseudo support samples, forming $K'$ distinct classes, i.e., $\mathcal{S}_{\text{val}} = \{(\mathbf{x}_k^s, y_k^s = k)\}_{k=1}^{K'}$. We regard the remaining unlabeled samples $\mathcal{Q}_{\text{val}} = \mathcal{D}_{\text{val}} \backslash \mathcal{S}_{\text{val}}$ as pseudo query samples. Next, pseudo-labels are generated for $\mathcal{Q}_{\text{val}}$ by applying instance-wise iterative prototype classification (Section 4.3) to the raw feature space, and those pseudo-labels serve as the "ground-truth" labels for the query set $\mathcal{Q}_{\text{val}}$. Subsequently, we employ D2R2 on the validation set $\{\mathcal{S}_{\text{val}}, \mathcal{Q}_{\text{val}}\}$ to predict the "ground-truth" labels of $\mathcal{Q}_{\text{val}}$. We assess the validation performance to select hyperparameters for the training iteration.

# 5 Experiments

## 5.1 Experimental settings

For all the datasets, we randomly split 80% of the data for training and the remaining 20% for testing. As for $N$-way $K$-shot scenario, the support set is constructed by randomly selecting $N \times K$ samples, with $K$ samples from each of the $N$ classes from the training sets. Additionally, 20% of the training data is utilized for validation and hyperparameter tuning. We use one-hot encoding for categorical features and min-max scaling for numerical features, except for income data, for which we use standardized scaling (see Appendix D.2 for details). All experiment settings of baselines and D2R2 are the same as STUNT [28] for fair comparison. Details are provided in Appendix D.3[2].

**Datasets**. We select nine datasets from the OpenML-CC18 benchmark [3, 6] to validate the performance of D2R2. Table 3 shows a summary of the datasets. Optdigits, mfeat-karhunen, diabetes contain and breast only numerical features; dna and mfeat-pixel consist of only categorical features; income, cmc and nomao contain both numerical and categorical features. A summary of the dataset information is provided in Appendix Table D.1.

**Baselines**. In order to assess the efficacy of our D2R2 framework, we compare the performance of D2R2 with four types of baselines, whose details are provided in Appendix E.

  i. Supervised learning methods. We compare with established supervised learning methods such as CatBoost [35], k-nearest neighbors (kNN) classifier [32] according to the prototype, and TabPFN [18], a transformer-based network designed to make predictions on a small tabular dataset.

 ii. Semi-supervised learning methods. Such methods include Mean Teacher(MT) [46], Interpolation Consistency Training (ICT) [48], Pseudo-Label [25] and Meta Pseudo Labels (MPL) [33].

iii. Few-shot meta-learning methods. We consider recent state-of-the-art meta-learning approaches such as CACTUs [19], UMTRA [22], SES [54] and STUNT [28]. STUNT is designed for tabular data while others are designed for image data, whose network structures are modified for tabular data[28].

 iv. Self-supervised learning methods. We compare with the state-of-the-art self-supervised methods for tabular data, including VIME[55], SubTab [47], SCARF [4], TabTransformer [20]. We utilize representations acquired from those models to conduct Center Prototype Classification [11].

## 5.2 Overall evaluation results

We compare our framework D2R2 with other state-of-the-art supervised, semi-supervised, self-supervised and meta-learning methods. The classification accuracy of all methods is presented in Table 1. In particular, we carry out experiments under two few-shot settings: $N$-way 1-shot and $N$-way 5-shot. We report the mean accuracy across 100 random seeds.

We note that our D2R2 framework significantly outperforms baseline methods across diverse datasets. A Wilcoxon signed-ranks test is employed (Appendix D.5) to further demonstrate the statistical significance of comparison. When comparing the accuracy of the D2R2 with that of the best baseline on all datasets, Wilcoxon's P-value is below 0.05, significantly indicating the effectiveness of D2R2.

In the case of the diabetes, optdigit and karkunen datasets, where all features are numerical, STUNT method outperforms other baselines, while D2R2 demonstrates substantial improvement. In the high-dimensional dataset optdigit, D2R2 improves accuracy by 39% compared to the supervised method Catboost, 23% compared to the semi-supervised method Mean Teacher, 5% compared to the meta-learning method STUNT, and 22% compared to other self-supervised methods like SubTab. On the other hand, for pixel and dna datasets, which only contain categorical features, meta-learning such as UMTRA and SES are almost ineffective, while D2R2 still demonstrates superior accuracy, outperforming all other methods in both one-shot and five-shot settings. Similarly, for the large size dataset income, encompassing both numerical and categorical features, our framework consistently achieves a high classification accuracy of 72.08%, surpassing the leading baseline STUNT by 19%. In summary, it can be concluded that D2R2 demonstrates a robust capability to address few-shot

Table 1: Reported test accuracy is the mean value across 100 random seeds. Asterisked (⋆) baselines refer to the reported scores in STUNT[28]. Bold number and the underlined number denote the highest score and the second best score, respectively. Empty data is either because the dataset exceeds the input data dimension limits of TabPFN or there are no reported scores in STUNT [28].

| Method | cmc | diabetes | dna | income | karkunen | optdigits | pixel | nomao | brest |
|---|---|---|---|---|---|---|---|---|---|
| #shot=1 | | | | | | | | | |
| CatBoost | 36.03 | 56.74 | 39.15 | 57.55 | 53.24 | 58.30 | 54.74 | 63.62 | 69.71 |
| TabPFN | 35.37 | 53.35 | - | - | 46.02 | 55.74 | - | - | - |
| KNN | 35.39 | 58.50 | 42.20 | 51.45 | 54.61 | 65.60 | 60.79 | 63.51 | 71.87 |
| Mean Teacher(*) | 35.58 | 58.05 | 46.58 | 60.63 | 54.57 | 66.10 | 61.02 | 64.23 | 71.92 |
| ICT(*) | 36.53 | 58.08 | 46.55 | 61.83 | 58.37 | 69.12 | 60.88 | - | - |
| Pseudo-Label(*) | 34.97 | 57.03 | 44.26 | 60.52 | 49.44 | 61.50 | 56.12 | 62.39 | 69.92 |
| MPL(*) | 35.13 | 57.39 | 44.22 | 60.85 | 47.66 | 61.52 | 56.01 | 64.28 | 71.33 |
| SubTab | 36.23 | 58.22 | 46.98 | 62.45 | 50.22 | 62.01 | 60.34 | 67.63 | 72.94 |
| VIME | 35.90 | 58.99 | 51.23 | 61.82 | 59.81 | 69.26 | 63.28 | 64.75 | 70.11 |
| SCARF | 35.39 | 55.64 | 57.86 | 57.94 | 60.96 | 63.31 | 63.93 | 68.90 | 75.32 |
| RTDL | 34.34 | 58.15 | 47.99 | 53.61 | 58.25 | 62.78 | 62.87 | 68.33 | 76.38 |
| UMTRA(*) | 35.46 | 57.64 | 25.13 | 57.23 | 49.05 | 49.87 | 34.26 | - | - |
| SES(*) | 34.59 | 59.97 | 39.56 | 56.39 | 49.19 | 56.30 | 49.19 | 69.52 | 74.89 |
| CACTUs(*) | 36.10 | 58.92 | 65.93 | 64.02 | 65.59 | 71.98 | 67.61 | 71.49 | 75.24 |
| STUNT(*) | 37.10 | 61.08 | 66.20 | 63.52 | 71.20 | 76.94 | 79.05 | 71.54 | 76.92 |
| D2R2 | **42.88** | **63.94** | **68.00** | **75.82** | **72.08** | **81.13** | **81.34** | **79.47** | **77.69** |
| #shot=5 | | | | | | | | | |
| CatBoost | 39.89 | 64.51 | 60.20 | 67.99 | 77.94 | 83.07 | 83.38 | 75.32 | 77.06 |
| TabPFN | 38.31 | 64.06 | - | - | 76.59 | 81.68 | - | - | - |
| KNN | 37.65 | 65.61 | 61.16 | 62.19 | 80.08 | 84.16 | 84.75 | 73.78 | 79.43 |
| Mean Teacher(*) | 37.73 | 65.45 | 61.47 | 67.05 | 81.08 | 86.66 | 85.24 | 74.78 | 81.26 |
| ICT(*) | 38.09 | 65.47 | 63.37 | 70.13 | 84.58 | 87.01 | 86.12 | - | - |
| Pseudo-Label(*) | 37.49 | 64.46 | 60.06 | 66.26 | 78.60 | 83.71 | 82.94 | 72.87 | 78.91 |
| MPL(*) | 37.47 | 64.51 | 59.65 | 67.61 | 77.85 | 83.70 | 82.39 | 73.20 | 79.54 |
| SubTab | 39.81 | 68.26 | 62.49 | 72.14 | 70.88 | 83.27 | 80.41 | 76.15 | 82.74 |
| VIME | 39.83 | 67.64 | 71.29 | 72.19 | 19.42 | 83.21 | 85.24 | 74.96 | 85.81 |
| SCARF | 37.75 | 68.66 | 62.75 | 66.09 | 69.96 | 85.67 | 81.32 | 77.65 | 84.42 |
| RTDL | 37.59 | 64.27 | 45.49 | 64.92 | 60.43 | 82.58 | 76.13 | 73.61 | 79.66 |
| UMTRA(*) | 38.05 | 64.41 | 25.08 | 65.78 | 67.28 | 73.29 | 51.32 | - | - |
| SES(*) | 39.04 | 66.61 | 52.25 | 68.27 | 74.80 | 78.46 | 74.80 | 76.50 | 84.73 |
| CACTUs(*) | 38.81 | 66.79 | 81.52 | 72.03 | 82.20 | 85.92 | 85.25 | 78.33 | 86.90 |
| STUNT(*) | 40.40 | 69.88 | 79.18 | 72.69 | **85.45** | 88.42 | 89.08 | 81.49 | 86.82 |
| D2R2 | **43.39** | **73.52** | **82.38** | **76.02** | 84.96 | **90.73** | **91.06** | **82.69** | **88.27** |

Table 2: Ablation study of three components: diffusion representation (DR), random distance matching (RDM) and instance-wise iterative prototype (IP). We also conduct the center-specific prototype version of D2R2 (D2R2-c).We report the test accuracy (%) under 100 random seeds.

| Dataset | RDM | DR | DR+RDM | RDM+IP | DR+IP | D2R2-c | D2R2 |
|---|---|---|---|---|---|---|---|
| opt (1-shot) | 49.05 | 72.38 | 77.41 | 26.66 | 76.87 | - | **81.13** |
| dna (1-shot) | 45.43 | 57.14 | 61.29 | 26.03 | 56.17 | - | **68.00** |
| cmc (1-shot) | 35.50 | 35.19 | 38.69 | 34.90 | 34.48 | - | **42.88** |
| opt (5-shot) | 70.26 | 88.64 | 89.61 | 51.31 | 89.32 | 87.12 | **90.73** |
| dna (5-shot) | 47.72 | 71.84 | 73.03 | 32.16 | 79.24 | 81.39 | **82.38** |
| cmc (5-shot) | 36.04 | 35.62 | 40.81 | 36.27 | 35.74 | 43.39 | **43.39** |

tabular data classification tasks, regardless of the number of shots, dataset size, feature dimension, or the proportion of categorical features.

We speculate that the superior performance of D2R2 is attributed to the effective learned representation of inputs along with the instance-specific prototypes. Upon visual examination of D2R2's embeddings in Figure 3, we observe clustering characteristics from point clouds, which demonstrating the embeddings' effectiveness for classification. Besides, we note the presence of multimodality in the embeddings, supporting the rationale for introducing the instance-wise prototypes.

## 5.3 Ablation study

We conduct an ablation study to demonstrate the efficacy of three components in D2R2, namely the diffusion representation (DR), random distance matching (RDM), and the instance-wise iterative prototype (IP). We compare the complete D2R2 design (DR+RDM+IP) with frameworks that exclude one or two of these components. We also conduct the center-specific prototype version of D2R2 (D2R2-c), which utilizes average embeddings of $K$-shot samples as prototype. The results of the ablation study are presented in the Table 2.

Based on the results, D2R2 surpasses all other variants, indicating that all three components collectively contribute to its superior performance. Specifically, removing the diffusion representation results in a significant degradation in performance, highlighting the crucial contribution of diffusion representation in capturing semantic knowledge. We also observe that DR+RDM shows the second best performance, highlighting the importance of incorporating random distance matching in the diffusion training process. Comparing with a center-specific prototype version of D2R2 (D2R2-c), we observe that the instance-wise prototypes leads to improvement results in optdigit and dna.

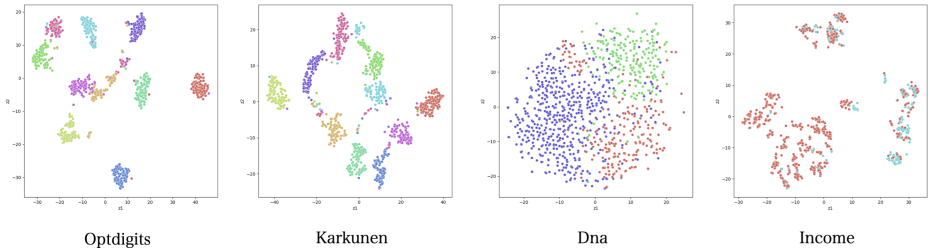

| Optdigits | Karkunen | Dna | Income |

Figure 3: The t-SNE visualizations depict the D2R2 representations, with point clouds illustrating the embeddings of 1000 randomly selected samples, color-coded based on their respective class labels. We observe that the embeddings exhibit multimodal patterns. For instance, in the dna dataset, the red class is distributed in both the bottom right corner and the top left corner.

## 6 Conclusion

In this paper, we introduce D2R2, an innovative framework to address few-shot tabular challenges. The core idea of D2R2 is to utilize the strong expressive abilities of diffusion model, along with a random distance matching to construct representation learners. This method captures the semantic knowledge of unlabeled data and generate effective embeddings for downstream classification tasks, meanwhile adapting to the mixture feature types of the tabular data. Additionally, to accommodate multimodalities of embeddings, we devise the instance-wise iterative prototype classifier using labeled data. Furthermore, a pseudo-label validation scheme is designed for hyper-parameter selection. The superior performance of D2R2 is presented on diverse datasets, demonstrating the effectiveness of D2R2 over other baselines. We hope that our work will inspire new avenues for research in the field of diffusion model representation learning on tabular datasets.

## 7 Acknowledgements

We would like to thank AC and reviewers for their valuable comments on the manuscript. Bingyi Jing's research is partly supported by NSFC 12371290.

## Footnotes

[2]Code available at `https://github.com/Carol-cloud-project/D2R2`

# References

[1] Korbinian Abstreiter, Sarthak Mittal, Stefan Bauer, Bernhard Schölkopf, and Arash Mehrjou. Diffusion-based representation learning. *arXiv preprint arXiv:2105.14257*, 2021.

[2] Sercan Ö Arik and Tomas Pfister. Tabnet: Attentive interpretable tabular learning. In *Proceedings of the AAAI conference on artificial intelligence*, volume 35, pages 6679–6687, 2021.

[3] Arthur Asuncion and David Newman. Uci machine learning repository, 2007.

[4] Dara Bahri, Heinrich Jiang, Yi Tay, and Donald Metzler. Scarf: Self-supervised contrastive learning using random feature corruption. *arXiv preprint arXiv:2106.15147*, 2021.

[5] Sungyong Baik, Janghoon Choi, Heewon Kim, Dohee Cho, Jaesik Min, and Kyoung Mu Lee. Meta-learning with task-adaptive loss function for few-shot learning. In *Proceedings of the IEEE/CVF international conference on computer vision*, pages 9465–9474, 2021.

[6] Bernd Bischl, Giuseppe Casalicchio, Matthias Feurer, Pieter Gijsbers, Frank Hutter, Michel Lang, Rafael Gomes Mantovani, Jan N van Rijn, and Joaquin Vanschoren. Openml benchmarking suites. In *Proceedings of the NeurIPS 2021 Datasets and Benchmarks Track*. 2021.

[7] Longbing Cao. Ai in finance: challenges, techniques, and opportunities. *ACM Computing Surveys (CSUR)*, 55(3):1–38, 2022.

[8] Ting Chen, Simon Kornblith, Mohammad Norouzi, and Geoffrey Hinton. A simple framework for contrastive learning of visual representations. In *International conference on machine learning*, pages 1597–1607. PMLR, 2020.

[9] Wei-Yu Chen, Yen-Cheng Liu, Zsolt Kira, Yu-Chiang Frank Wang, and Jia-Bin Huang. A closer look at few-shot classification. *arXiv preprint arXiv:1904.04232*, 2019.

[10] Xinlei Chen, Zhuang Liu, Saining Xie, and Kaiming He. Deconstructing denoising diffusion models for self-supervised learning. *arXiv preprint arXiv:2401.14404*, 2024.

[11] Yudong Chen, Chaoyu Guan, Zhikun Wei, Xin Wang, and Wenwu Zhu. Metadelta: A meta-learning system for few-shot image classification. In *AAAI Workshop on Meta-Learning and MetaDL Challenge*, pages 17–28. PMLR, 2021.

[12] Zitian Chen, Subhransu Maji, and Erik Learned-Miller. Shot in the dark: Few-shot learning with no base-class labels. In *Proceedings of the IEEE/CVF Conference on Computer Vision and Pattern Recognition*, pages 2668–2677, 2021.

[13] Hao Cheng, Joey Tianyi Zhou, Wee Peng Tay, and Bihan Wen. Graph neural networks with triple attention for few-shot learning. *IEEE Transactions on Multimedia*, 2023.

[14] Hongchao Fang, Sicheng Wang, Meng Zhou, Jiayuan Ding, and Pengtao Xie. Cert: Contrastive self-supervised learning for language understanding. *arXiv preprint arXiv:2005.12766*, 2020.

[15] Chelsea Finn, Pieter Abbeel, and Sergey Levine. Model-agnostic meta-learning for fast adaptation of deep networks. In *International conference on machine learning*, pages 1126–1135. PMLR, 2017.

[16] Kaiming He, Haoqi Fan, Yuxin Wu, Saining Xie, and Ross Girshick. Momentum contrast for unsupervised visual representation learning. In *Proceedings of the IEEE/CVF Conference on Computer Vision and Pattern Recognition*, pages 9729–9738, 2020.

[17] Jonathan Ho, Ajay Jain, and Pieter Abbeel. Denoising diffusion probabilistic models. *Advances in neural information processing systems*, 33:6840–6851, 2020.

[18] Noah Hollmann, Samuel Müller, Katharina Eggensperger, and Frank Hutter. Tabpfn: A transformer that solves small tabular classification problems in a second. *arXiv preprint arXiv:2207.01848*, 2022.

[19] Kyle Hsu, Sergey Levine, and Chelsea Finn. Unsupervised learning via meta-learning. *arXiv preprint arXiv:1810.02334*, 2018.

[20] Xin Huang, Ashish Khetan, Milan Cvitkovic, and Zohar Karnin. Tabtransformer: Tabular data modeling using contextual embeddings. *arXiv preprint arXiv:2012.06678*, 2020.

[21] Dahyun Kang, Heeseung Kwon, Juhong Min, and Minsu Cho. Relational embedding for few-shot classification. In *Proceedings of the IEEE/CVF International Conference on Computer Vision*, pages 8822–8833, 2021.

[22] Siavash Khodadadeh, Ladislau Boloni, and Mubarak Shah. Unsupervised meta-learning for few-shot image classification. *Advances in neural information processing systems*, 32, 2019.

[23] Diederik P Kingma and Jimmy Ba. Adam: A method for stochastic optimization. *arXiv preprint arXiv:1412.6980*, 2014.

[24] Dong Bok Lee. Meta-gmvae: Mixture of gaussian vaes for unsupervised meta-learning. 2021.

[25] Dong-Hyun Lee et al. Pseudo-label: The simple and efficient semi-supervised learning method for deep neural networks. In *Workshop on challenges in representation learning, ICML*, volume 3, page 896. Atlanta, 2013.

[26] Junjie Li, Zilei Wang, and Xiaoming Hu. Learning intact features by erasing-inpainting for few-shot classification. In *Proceedings of the AAAI conference on artificial intelligence*, volume 35, pages 8401–8409, 2021.

[27] Mario Molina and Filiz Garip. Machine learning for sociology. *Annual Review of Sociology*, 45:27–45, 2019.

[28] Jaehyun Nam, Jihoon Tack, Kyungmin Lee, Hankook Lee, and Jinwoo Shin. Stunt: Few-shot tabular learning with self-generated tasks from unlabeled tables. *arXiv preprint arXiv:2303.00918*, 2023.

[29] Alex Nichol, Joshua Achiam, and John Schulman. On first-order meta-learning algorithms. *arXiv preprint arXiv:1803.02999*, 2018.

[30] Jaehoon Oh, Hyungjun Yoo, ChangHwan Kim, and Se-Young Yun. Boil: Towards representation change for few-shot learning. *arXiv preprint arXiv:2008.08882*, 2020.

[31] Ethan Perez, Douwe Kiela, and Kyunghyun Cho. True few-shot learning with language models. *Advances in neural information processing systems*, 34:11054–11070, 2021.

[32] Leif E Peterson. K-nearest neighbor. *Scholarpedia*, 4(2):1883, 2009.

[33] Hieu Pham, Zihang Dai, Qizhe Xie, and Quoc V Le. Meta pseudo labels. In *Proceedings of the IEEE/CVF conference on computer vision and pattern recognition*, pages 11557–11568, 2021.

[34] Mihir Prabhudesai, Shamit Lal, Darshan Patil, Hsiao-Yu Tung, Adam W Harley, and Katerina Fragkiadaki. Disentangling 3d prototypical networks for few-shot concept learning. *arXiv preprint arXiv:2011.03367*, 2020.

[35] Liudmila Prokhorenkova, Gleb Gusev, Aleksandr Vorobev, Anna Veronika Dorogush, and Andrey Gulin. Catboost: unbiased boosting with categorical features. *Advances in neural information processing systems*, 31, 2018.

[36] Guo-Jun Qi and Jiebo Luo. Small data challenges in big data era: A survey of recent progress on unsupervised and semi-supervised methods. *IEEE Transactions on Pattern Analysis and Machine Intelligence*, 44(4):2168–2187, 2020.

[37] Alec Radford, Jeffrey Wu, Rewon Child, David Luan, Dario Amodei, Ilya Sutskever, et al. Language models are unsupervised multitask learners. *OpenAI blog*, 1(8):9, 2019.

[38] Mengye Ren, Eleni Triantafillou, Sachin Ravi, Jake Snell, Kevin Swersky, Joshua B Tenenbaum, Hugo Larochelle, and Richard S Zemel. Meta-learning for semi-supervised few-shot classification. *arXiv preprint arXiv:1803.00676*, 2018.

[39] Eli Schwartz, Leonid Karlinsky, Rogerio Feris, Raja Giryes, and Alex Bronstein. Baby steps towards few-shot learning with multiple semantics. *Pattern Recognition Letters*, 160:142–147, 2022.

[40] K Shailaja, Banoth Seetharamulu, and MA Jabbar. Machine learning in healthcare: A review. In *2018 Second international conference on electronics, communication and aerospace technology (ICECA)*, pages 910–914. IEEE, 2018.

[41] Jake Snell, Kevin Swersky, and Richard Zemel. Prototypical networks for few-shot learning. *Advances in neural information processing systems*, 30, 2017.

[42] Jascha Sohl-Dickstein, Eric Weiss, Niru Maheswaranathan, and Surya Ganguli. Deep unsupervised learning using nonequilibrium thermodynamics. In *International conference on machine learning*, pages 2256–2265. PMLR, 2015.

[43] Kihyuk Sohn, David Berthelot, Nicholas Carlini, Zizhao Zhang, Han Zhang, Colin A Raffel, Ekin Dogus Cubuk, Alexey Kurakin, and Chun-Liang Li. Fixmatch: Simplifying semi-supervised learning with consistency and confidence. *Advances in neural information processing systems*, 33:596–608, 2020.

[44] Yang Song and Stefano Ermon. Generative modeling by estimating gradients of the data distribution. *Advances in neural information processing systems*, 32, 2019.

[45] Yang Song, Jascha Sohl-Dickstein, Diederik P Kingma, Abhishek Kumar, Stefano Ermon, and Ben Poole. Score-based generative modeling through stochastic differential equations. *arXiv preprint arXiv:2011.13456*, 2020.

[46] Antti Tarvainen and Harri Valpola. Mean teachers are better role models: Weight-averaged consistency targets improve semi-supervised deep learning results. *Advances in neural information processing systems*, 30, 2017.

[47] Talip Ucar, Ehsan Hajiramezanali, and Lindsay Edwards. Subtab: Subsetting features of tabular data for self-supervised representation learning. *Advances in Neural Information Processing Systems*, 34:18853–18865, 2021.

[48] Vikas Verma, Kenji Kawaguchi, Alex Lamb, Juho Kannala, Arno Solin, Yoshua Bengio, and David Lopez-Paz. Interpolation consistency training for semi-supervised learning. *Neural Networks*, 145:90–106, 2022.

[49] Hu Wang, Guansong Pang, Chunhua Shen, and Congbo Ma. Unsupervised representation learning by predicting random distances. *arXiv preprint arXiv:1912.12186*, 2019.

[50] Yaqing Wang, Quanming Yao, James T Kwok, and Lionel M Ni. Generalizing from a few examples: A survey on few-shot learning. *ACM computing surveys (csur)*, 53(3):1–34, 2020.

[51] Jiangtao Xie, Fei Long, Jiaming Lv, Qilong Wang, and Peihua Li. Joint distribution matters: Deep brownian distance covariance for few-shot classification. In *Proceedings of the IEEE/CVF conference on computer vision and pattern recognition*, pages 7972–7981, 2022.

[52] Chen Xing, Negar Rostamzadeh, Boris Oreshkin, and Pedro O O Pinheiro. Adaptive cross-modal few-shot learning. *Advances in Neural Information Processing Systems*, 32, 2019.

[53] Xingyi Yang and Xinchao Wang. Diffusion model as representation learner. In *Proceedings of the IEEE/CVF International Conference on Computer Vision*, pages 18938–18949, 2023.

[54] Han-Jia Ye, Lu Han, and De-Chuan Zhan. Revisiting unsupervised meta-learning via the characteristics of few-shot tasks. *IEEE Transactions on Pattern Analysis and Machine Intelligence*, 45(3):3721–3737, 2022.

[55] Jinsung Yoon, Yao Zhang, James Jordon, and Mihaela van der Schaar. Vime: Extending the success of self-and semi-supervised learning to tabular domain. *Advances in Neural Information Processing Systems*, 33:11033–11043, 2020.

[56] Baoquan Zhang, Xutao Li, Yunming Ye, Zhichao Huang, and Lisai Zhang. Prototype completion with primitive knowledge for few-shot learning. In *Proceedings of the IEEE/CVF Conference on Computer Vision and Pattern Recognition*, pages 3754–3762, 2021.

# A Algorithm

---

**Algorithm 1** Training D2R2

---

**Require:** Unlabeled dataset $\mathcal{D}_u = \{\mathbf{x}_i\}_{i=1}^{N_u}$, perturbation scale $\sigma$, batch size $B$, diffusion horizon $T$, noise schedule $\{\beta_t\}_{t=1}^{T}$, $\{\alpha_t\} = \{1 - \beta_t\}$, $\{\bar{\alpha}_t\} = \{\prod_{t'=1}^{t} \alpha_{t'}\}$, number of numerical features $d_{\text{num}}$, number of categorical features $d_{\text{cat}}$, feature dimension $d = d_{\text{num}} + d_{\text{cat}}$, learning rates $\gamma_1$, $\gamma_2$, hyperparameters: $\alpha, p, A$.

---

**Initialize:** noisy network $\epsilon_\phi$, embedding network $z_\theta$
    **while** not done **do**
        Sample two batchs of unlabeled data $\mathbf{x}, \mathbf{x}' \sim \mathcal{D}_u$.
        Sample a perturbation copy $\tilde{\mathbf{x}} \sim x + \sigma \mathcal{N}(0, 1)$.
        Sample timestep $t \sim 1, 2, ..., T$ and noise $\epsilon \sim \mathcal{N}(0, 1)$.
        Compute the noisy sample $\mathbf{x}_t = \sqrt{\bar{\alpha}_t} \mathbf{x} + \sqrt{1 - \bar{\alpha}_t} \epsilon$.
        $\mathcal{L}_{\text{recon}}(\phi, \theta) = ||\epsilon_\phi(\mathbf{x}_t, t, z_\theta(\tilde{\mathbf{x}})) - \epsilon||_2^2$.
        Sample matrix $W_n \in \mathbb{R}^{d \times r_1}$ with $(W_n)_{ij} \sim_{i.i.d.} \text{Unif}(-A, A)$.
        Sample matrix $W_c \in \mathbb{R}^{d \times r_2}$ with $(W_c)_{ij} \sim_{i.i.d.} \text{Bernoulli}(p)$.
        Compute $W\mathbf{x} = \text{Concat}(W_n[\mathbf{x}]_{\text{num}}, W_c[\mathbf{x}]_{\text{cat}})$,
        Compute $W\mathbf{x}' = \text{Concat}(W_n[\mathbf{x}']_{\text{num}}, W_c[\mathbf{x}']_{\text{cat}})$,
        $\mathcal{L}_{\text{rdm}}(\theta) = ||\text{cosine}(z_\theta(\mathbf{x}), z_\theta(\mathbf{x}')) - \text{cosine}(W\mathbf{x}, W\mathbf{x}')||^2$,
        $\phi \leftarrow \phi - \frac{\gamma_1}{B} \cdot \nabla_\phi \mathcal{L}_{\text{recon}}(\phi, \theta)$.
        $\theta \leftarrow \theta - \frac{\gamma_2}{B} \cdot [\nabla_\theta \mathcal{L}_{\text{recon}}(\phi, \theta) + \alpha \nabla_\theta \mathcal{L}_{\text{rdm}}(\theta)]$.
    **end while**
        **return** Embedding function $z_\theta$.

---

**Algorithm 2** Pseudo-label Validation

---

**Require:** Unlabeled validation set $\mathcal{D}_{\text{val}} = \{\mathbf{x}_i\}_{i=1}^{N_v}$, number of pseudo-classes $K'$, number of iteration $I$

---

    **for** $i = 1, 2, ...I$ **do**
        Sample $K'$ points $\{\mathbf{x}_i\}_{i=1}^{K'} \in \mathcal{D}_{\text{val}}$.
        Form support set $\mathcal{S} = \{(\mathbf{x}_i, i)\}_{i=1}^{K'}$.
        Form query set $\mathcal{Q} = \mathcal{D}_{\text{val}} \setminus \{\mathbf{x}_i\}_{i=1}^{K'}$.
        Create instance-wise iterative prototype classifier $f(\cdot|, \mathcal{S}, \mathcal{Q})$ using the raw feature vectors.
        Create pseudo-label in $\mathcal{Q} \leftarrow \{(\mathbf{x}_i, f(\mathbf{x}_i|\mathcal{S}))|\mathbf{x}_i \in \mathcal{Q}\}$.
    **end for**
        **return** Pseudo support set $\mathcal{S}$, pseudo query set $\mathcal{Q}$

---

# B Diffusion Models

Diffusion-based generative models [42, 17, 44, 45] are latent variable models that use a Markovian noising and parameterized denoising process to model the data distribution. The forward noising process of horizon $T$ follows a pre-defined variance schedule $\beta_1, ..., \beta_T$ that encodes data distribution $\mathbf{x}_0 \sim p(x)$:

$$q(\mathbf{x}_t|\mathbf{x}_{t-1}) = \mathcal{N}(\sqrt{1 - \beta_t}\mathbf{x}_{t-1}, \beta_t \mathbf{I}), \tag{9}$$

and assume the prior $q(\mathbf{x}_t) \sim \mathcal{N}(0, \mathbf{I})$. The reverse denoising process is modeled as $p_\phi(\mathbf{x}_{0:T}) := \mathcal{N}(\mathbf{x}_t; 0, \mathbf{I}) \prod_{t=1}^{T} p_\phi(\mathbf{x}_{t-1}|\mathbf{x}_t)$ with learnable parameters $\phi$. Following DDPMs [17], the denoising model can be implemented by neural networks directly predicting the noise $\epsilon$:

$$\mathcal{L}_{\text{DDPM}}(\phi) = \mathbf{E}_{t, \mathbf{x}_0} ||\epsilon_\phi(\mathbf{x}_t, t) - \epsilon||_2^2, \tag{10}$$

where $\mathbf{x}_t = \sqrt{\bar{\alpha}_t}\mathbf{x}_0 + \sqrt{1 - \bar{\alpha}_t}\epsilon, \epsilon \sim \mathcal{N}(0, \mathbf{I})$ and $\bar{\alpha}_t = \prod_{t'=1}^{t}(1 - \beta'_t)$. Diffusion models can be straightforwardly extended to conditional generative models $\epsilon_\phi(\mathbf{x}_t, t, c)$ by inserting the conditional information $c$ [45, 1].

## C   Random Linear Projection

We generate random matrices from different distributions to cater for the numerical features and categorical features separately. In order to integrate the projections derived from numerical and categorical spaces, it is necessary to select $A$ (for Uniform$(-A, A)$) and $p$ (for Bernoulli$(p)$) in a consistent manner to equitably consider the information from heterogeneous type. In this paper, we adjust the random weights according to the consistent activation variance, that is we set $A = \sqrt{3p(1-p)}$ and treat $p$ as a hyperparameter, and re-weight the matrices according to the number of columns, i.e., $W_{num} \leftarrow W_{num}/\sqrt{d_{num}}, W_{cat} \leftarrow W_{cat}/\sqrt{d_{cat}}$. Such choice of parameters preserve the variance of the row-sum of projection matrices . Additionally, we sample $W_{num} \in \mathbb{R}^{d_{num} \times d_{num}}$ and $W_{cat} \in \mathbb{R}^{d_{cat} \times d_{cat}}$ to get rid of the need to treat $r_1, r_2$ in Algorithm (1) as hyperparameters. We leave the study of different random matrices to the future work.

## D   Experiment details

### D.1   Dataset Information

Table 3: Datasets information. The table shows the instances number, the number of features (numerical features, categorical features) and the number of classes of each dataset.

| Dataset | # Instances | # Features (num., cate.) | # Classes |
|---------|-------------|--------------------------|-----------|
| optdigit | 5620 | 64 (64,0) | 10 |
| karkunen | 2000 | 64 (64,0) | 10 |
| diabetes | 768 | 8 (8,0) | 2 |
| pixel | 2000 | 240 (0,240) | 2 |
| dna | 3186 | 180 (0,180) | 3 |
| cmc | 1473 | 9 (2,7) | 3 |
| income | 48842 | 14 (6,8) | 2 |
| nomao | 34465 | 118 (90,28) | 2 |
| brest | 286 | 24481(24481,0) | 2 |

### D.2   Data normalization

For income dataset, we apply standardized scaling to each numerical feature $x_i$:

$$x_i \leftarrow \frac{x_i - \mu(x_i)}{\sigma(x_i)}. \tag{11}$$

And we employ min-max scaling numerical features of other datasets:

$$x_i \leftarrow \frac{x_i - \min(x_i)}{\max(x_i) - \min(x_i)}. \tag{12}$$

### D.3   Hyperparameter Details

We employ fully connected layers with identical layer counts and hidden dimensions to model the diffusion noise and the embedding function. The training procedure for the diffusion model follows the same approach as in DDPM [17]. The configuration for the diffusion model and neural networks (NN) are as following:

In our experiments, we search three hyperparameters during the validation process.

All other experiment settings of D2R2 and other baseline are the same as STUNT [28] for fair comparison.

### D.4   Choosing Noise Levels

The timestep $t = 1, ..., T$ indicates noise levels in diffusion models. In our experimental analysis, we have noted a trend where performance sees an improvement with an increase in $t$ during the initial

Table 4: Model configurations.

| Parameter | Setting | Description |
|---|---|---|
| T | 10 | Diffusion timesteps. |
| $\beta_t$ | vp [45] | Noise schedule. |
| Hidden dims | 512 | Dimension of dense layers. |
| Layers | 3 | Number of dense layers. |
| Optimizer | Adam [23] | - |
| Batch Size | 256 | - |
| Learning Rate | 0.0003 | - |

Table 5: Hyperparameter search range

| Parameter | Range | Description |
|---|---|---|
| $\alpha$ | linespace(0.1, 0.1, 5) | RDM weight. |
| $d_z$ | $\{5, 10, 20, 40, 80\}$ | Embedding dimension |
| $\tau$ | linespace(0.1, 0.1, 2) | Temperature in equ. (6) |

stages. However, this trend of performance enhancement plateaus after the first few timesteps, which tends to stabilize with minimal fluctuations, as shown in Table 6. As $t$ approaches $T$, noise levels rise, thus effective denoising requires that all information about $x_0$ be thoroughly encoded, which contains the most information about the data class. We focus on larger timesteps thus extract the low-frequency semantic information rather than details. Thus in our experiments, we selected the last step $T$ for all datasets.

Table 6: Performance Across Different Timesteps

| Dataset | Shot | First Step | Middle Step | 80th percentile Step | Last Step | Average of All Steps |
|---|---|---|---|---|---|---|
| optdigits | 1 | 72.39 | 79.21 | 80.93 | **81.13** | 80.66 |
| optdigits | 5 | 84.57 | 88.67 | **91.73** | 90.73 | 89.66 |
| cmc | 1 | 37.19 | 42.26 | 42.18 | **42.88** | 42.74 |
| cmc | 5 | 36.28 | 42.73 | 42.89 | **43.39** | 42.37 |

## D.5 Wilcoxon Test

The Wilcoxon signed-ranks test is a statistical method used to compare the means of two related groups when assumptions of normal distribution are not met. Particularly useful for paired data, it involves ranking the absolute differences between paired observations and calculating a test statistic for assessing significance. The procedure includes the following:

i. Calculate the differences: $D_i = X_i - Y_i$, where $X_i$ and $Y_i$ are paired observations.

ii. Rank $|D_i|$ from the smallest to the largest, resolving ties by averaging ranks.

iii. Calculate signed ranks: Assign positive ranks to positive differences and negative ranks to negative differences.

iv. Calculate the test statistic $W$, which is the sum of the positive or negative ranks, depending on which sum is smaller.

v. Compare $W$ to critical values: Determine significance by comparing $W$ to critical values from the Wilcoxon signed-ranks table or using statistical software.

In summary, the test helps identify if a significant difference exists between the groups' means, even when data doesn't meet assumptions for parametric tests.

## D.6 Model Complexity

The model complexity is manageable through our framework's utilization of two neural networks: the embedding network and the noise prediction network, both of which are demonstrated as 3-layer MLPs in our paper. These settings (hidden dimensions, embedding dimensions and model structure)

can be adjusted to meet different needs. In terms of computational efficiency, The main time-consuming factor after the embedding model is the calculation of the instance-wise iterative prototype. For an $N$-way $K$-shot problem with $L$ iterations and a query set of size $Q$, the computation complexity is $O(NKLQ)$. In few-shot learning, where $N, K, L$ are small, the computation complexity is linear in the size of the query set.

## E  Baseline Details

we compare with supervised learning methods such as CatBoost [35], k-nearest neighbors (kNN) [32], which denotes the nearest neighbor classifier according to the prototype of the input data, and TabPFN [18], which is a transformer that that solves small tabular classification problems.

Mean Teacher (MT) [46] is a semi-supervised learning framework designed to enhance model generalization by establishing a robust interaction between labeled and unlabeled samples. This framework draws inspiration from the "teacher-student" training approach, where the "teacher" refers to a model averaging multiple student model predictions, while the "student" is the model being trained.

Interpolation Consistency Training (ICT) [48] uses MT framework. While ICT aims to improve the model performance by augmenting additional data points through interpolation, with the expectation that the model's predicted values align consistently with the interpolated labels.

Meta Pseudo Labels [33] continually adjusts the teacher model based on feedback from the student model's performance on labeled datasets. Both teacher and student models are trained concurrently, with the teacher model learning to generate improved pseudo-labels while the student model learns from these pseudo-labels.

VIME[55] is a form of self-supervised learning that derives valuable representations by intentionally corrupting random features and subsequently predicting the corrupted location. We utilize VIME representations for conducting k-nearest neighbor classification.

SubTab [47] employs three effective pretext task losses, namely, reconstruction loss, contrastive loss, and distance loss, as part of its self-supervised learning methodology. We utilize SubTab representations for conducting k-nearest neighbor classification.

SCARF [4] takes a SimCLR-like [8] contrastive loss between the sample and its corrupted version. We utilize SubTab representations for conducting k-nearest neighbor classification.

TabTransformer [20] is built upon self-attention based Transformers. The Transformer layers transform the embeddings of categorical features into robust contextual embeddings to achieve high prediction accuracy in tabular context. Experiments shows that TabTransformer performs better than TabNet [2]. We utilize TabTransformer representations for conducting k-nearest neighbor classification.

CACTUs [19] creates episodic tasks by dividing the features extracted from an earlier-trained unsupervised feature embedding network using various objective functions, and then trains the few-shot learner on these tasks.

UMTRA [22] employs a domain-specific data augmentation approach to create synthetic tasks for the meta-learning stage. In doing so, the episodic tasks that are formed are constrained by the data augmentation strategy.

SES [54] considers the augmented sample as part of the same pseudo-class and implements robust strategies for unsupervised meta-learning, which include ample episodic sampling, challenging mixed supports, and task-specific projection heads.

## F  Limitations

Our approach is tailored to the specific characteristics of tabular data, including data scarcity, multiple modalities, mixed feature types, and sensitivity to changes in column order. Thus it is only effective in scenarios involving tabular data, and may not work well for CV and NLP tasks.

